# ART2/BP architecture for adaptive estimation of dynamic processes

**Einar Sørheim**[*]
Department of Computer Science
UNIK, Kjeller
University of Oslo
N-2007 Norway

## Abstract

The goal has been to construct a supervised artificial neural network that learns incrementally an unknown mapping. As a result a network consisting of a combination of ART2 and backpropagation is proposed and is called an "ART2/BP" network. The ART2 network is used to build and focus a supervised backpropagation network. The ART2/BP network has the advantage of being able to dynamically expand itself in response to input patterns containing new information. Simulation results show that the ART2/BP network outperforms a classical maximum likelihood method for the estimation of a discrete dynamic and nonlinear transfer function.

## 1 INTRODUCTION

Most current neural network architectures such as backpropagation require a cyclic presentation of the entire training set to converge. They are thus not very well suited for adaptive estimation tasks where the training vectors arrive one by one, and where the network may never see the same training vector twice. The ART2/BP network system is an attempt to construct a network that works well on these problems.

Main features of our ART2/BP are :

- implements incremental supervised learning
- dynamically self-expanding

---

[*]e-mail address: einar@tellus.unik.no or einars@ifi.uio.no

- learning of a novel training pattern does not wash away memory of previous training patterns
- short convergence time for learning a new pattern

## 2    BACKGROUND

Adaptive estimation of nonlinear functions requires some basic features of the estimation algorithm.

1. *Incremental learning*
   The input/output pairs arrive to the estimation machine one by one. By accumulating the input/output pairs into a training set and rerun the training procedure at every arrival of a new input/output pair, one could use a conventional method. Obvious disadvantages would however be
   - huge learning time required as the size of the training set increases.
   - an upper limit, N, on the number of elements in the training set will have to be set. The training set will then be a gliding horizon of the N last input/output pairs, and information prior to the N last input/output pairs will be lost.

2. *Plasticity*
   Learning of a new input/output pair should not wash away the memory of previously learned nonconflicting input/output pairs. With most existing feedforward supervised nets this is hard to accomplish, though some efforts have been made (Otwell 90). Some networks, like the ART-family and RCN (Ryan 1988) are plastic but they are self-organizing, not supervised.

To summarize:
Need a supervised network that learns incrementally the mapping of an unknown system and that can be used to predict future outputs. The system in question maps analog vectors to analog vectors.

## 3    COMBINED ARCHITECTURE

In the proposed network architecture an ART2 network controls a BP network, see Figure 1.

The BP-network consists of many relatively small subnetworks where the subnets are specialized on one particular domain of the input space. ART2 controls how the input space is divided among the subnets and the total amount of subnets needed.

The ART2 network analyzes the input part of the input/output pairs as they arrive to the system. For a given input pattern $\vec{i}_x$, ART2 finds the category $C_x$ which has the closest resemblance to $\vec{i}_x$. If this resemblance is good enough, $\vec{i}_x$ is of category $C_x$ and the LTM-weights of $C_x$ are updated. The BP-subnetwork $BP_x$, connected to $C_x$, is as a consequence activated and relearning of $BP_x$ is done. The learning set consists of a "representative" set of the neighbouring subnets patterns and a small number of the previous patterns belonging to category $C_x$. To summarize the

algorithm goes as follows:

1. Send input vector to ART2 network
2. ART2 classification.
3. If in learning mode adjust ART2 LTM weights of the winning node.
4. Send input to the back propagation network connected to the winning ART2 node.
5. If in learning mode :
   - find a representative training set.
   - do epoch learning on training set.

   Otherwise

   - compute output of the selected back propagation network.
6. Go to 1. for new input vector.

The ART2/BP neural network can be used for adaptive estimation of nonlinear dynamic processes. The mapping to be estimated then is

$$\vec{y}(t + \delta t) \quad = \quad \vec{f}(\vec{u}(t), \vec{y}(t)) \tag{1}$$
$$\vec{u}(t) \quad \epsilon \; \Re^m$$
$$\vec{y}(t) \quad \epsilon \; \Re^n$$

The input/output pairs will be $\vec{io} = [\vec{u}(t), \vec{y}(t), \vec{y}(t + \delta t)]$, denote the input part of $\vec{io}$: $\vec{i} = [\vec{u}(t), \vec{y}(t)]$ and the output part of $\vec{io}$: $\vec{o} = \vec{y}(t + \delta t)$.

## 4   ART2 MODIFIED

ART2 was developed by Carpenter& Grossberg see (Carpenter 1987) and (Carpenter 1988). ART2 categorizes arbitrary sequences of analog input patterns, and the categories can be of arbitrary coarseness. For a detailed description of ART2, see (Carpenter 1987).

### 4.1   MODIFICATION

In the standard ART2-algorithm input vectors (patterns) are normalized. For this application it is not desired to classify parallel vectors of different magnitude as belonging to the same category. By adding an extra element to the input vector where this element is simply

$$i_{n+1} = \|\vec{i}\|^2 \tag{2}$$

the new input vector becomes

$$\tilde{\vec{i}} = [\vec{i}, \|\vec{i}\|^2] \tag{3}$$

From a scaled vector of $\tilde{\vec{i}}$ : $\vec{x} = a \cdot \tilde{\vec{i}}$ the original vector $\vec{i}$ could easily be found as :

$$i_i \quad = \quad x_i \cdot \frac{x_{n+1}}{\|\tilde{\vec{x}}\|^2} \tag{4}$$
$$\tilde{\vec{x}} \quad = \quad [x_1, x_2, ...., x_n]$$

and by using the augmented $\tilde{\vec{i}}$ as the input to ART2 instead of $\vec{i}$ one can at any point in F1( representation layer ) and F2( categorization layer ) generate the corresponding non-normalized vector. The F2 node competition is modified so that the node having bottom-up LTM weights with the smallest distance (distance being the euclidean norm) to the F1 layer pattern code wins the competition. The distance $d_J$ of F2 node $J$ is given by:

$$
\begin{aligned}
d_J &= \|\vec{p} - \vec{z_J}\| \\
\| \ \| &: \quad being\ the\ l_2 - norm \\
\vec{p} &: \quad F1\ pattern\ code. \\
\vec{z_J} &: \quad bottom - up\ LTM\ weights\ of\ F2\ node\ J
\end{aligned}
\tag{5}
$$

Reset is done by calculating the distance $d$ between the F1 layer pattern code $\vec{p}$ and $\vec{i}$ :

$$
d = \|\vec{p} - \vec{i}\|
\tag{6}
$$

and comparing it to a largest acceptable bound $\rho$ . If $d > \rho$ the winning node is inhibited and a new node will be created. If $d \leq \rho$ LTM-patterns of the winning node $J$ are modified (learning).

# 5   BACK PROPAGATION NETWORK

The backpropagation network used in this work is of the standard feedforward type, see (Rumelhart 1986) . The number of hidden layers and nodes should be kept low in the subnetworks, for the problems in our simulations we used 1 hidden layer with 2 nodes. As for training algorithms several different kinds have been tried:

- Standard back propagation (SBP)

- A modified back propagation (MBP) method similar to the one used in the BPS simulator from George Mason University.

- Quickprop (Q).

- A quasi-Newton method (BFGS).

All of these except SBP show similar performance in my test cases.

The BP-networks performs as an interpolator in this algorithm and any good interpolation algorithm can be used instead of BP. Approximation theory gives several interesting techniques for approximation/interpolation of multidimensional functions such as Radial Basis Functions and Hyper Basis Functions, for further detail see (Poggio 90). These methods requires a representative training set where the input part determines the location of centers in the input space. The ART2 algorithm can be used for determining these centers in an adaptive way and thus making possible an incremental version of the approximation theory techniques. This idea has not been tested yet, but is an interesting concept for further research.

# 6   LEARNING

Learning in ART2/BP is a two stage process. First the input patterns is sent to the ART2 network for categorizing and learning . ART2 will then activate the BP subnetwork that is a local expert on patterns of the same category as the input pattern, and learning of this subnetwork will occur. A training set that is representative for the domain of the input space has to be found. Let a small number of the last categorized input/output pairs be allocated to its corresponding subnet to provide a part of the training set. Denote such a set as $L\_IO_C$, ( $C$ being the category). Define the location of F2 node $J$ to be its bottom-up weights $\vec{z}_J$. Let the current input $\vec{i}_x$ define an origin, then find the F2 nodes closest to origin in each n-ant of the input space. Call this set of nodes $N_x$ and the set of last input/output pairs stored in these nodes $N\_IO_x$. The training set is then chosen to be:
$T_x = N\_IO_x \cup L\_IO_x$
Before training, the elements in $T_x$ are scaled to increase accuracy and to accelerate learning. BP-learning is then performed, the stopping criteria being a fixed error term or a maximum number of iterations.

# 7   ESTIMATION

In estimation mode learning in the network is turned off. Given an input thenetwork will produce an output that hopefully will be close to the output of the real system.

The ART2-network selects a winning node in the same way as described before but now the reset assembly is not activated. Then the input is fed to the corresponding BP subnetwork and its output is used as an estimate of the original functions output.

Because each subnetwork is scaled to cover the domain of the input space made up by the complex hull $Co(T_x)$ of its training set $T_x$, the entire ART2/BP network will cover the complex hull $Co(T) \subset \Re^{n+m}$ where:
$T =$
$\{set\ of\ all\ previous\ \vec{i}'s\ used\ to\ train\ the\ network\}$
Good estimation/prediction can thus be expected if $\vec{i} \in Co(T)$. This means that if the input vector $\vec{i}$ lies in a domain of the input space that has not been previously explored by the elements in the training set, the network will generalize poorly.

# 8   EXAMPLE

The ART2/BP network has been used to estimate a dynamic model of a tank filled with liquid. The liquid level is sampled every $\delta t$ time interval and the ART2/BP network is used to estimate the discrete dynamic nonlinear transfer function of the liquid level as a function of inlet liquid flow and previous liquid level. That is, we want to find a good estimate $\hat{f}(\cdot, \cdot)$ of:

$$
\begin{aligned}
y(t + \delta t) &= f(u(t), y(t)) && (7)\\
u(t) &= inlet\ liquid\ flow\ at\ time\ t\\
y(t) &= liquid\ level\ at\ time\ t
\end{aligned}
$$

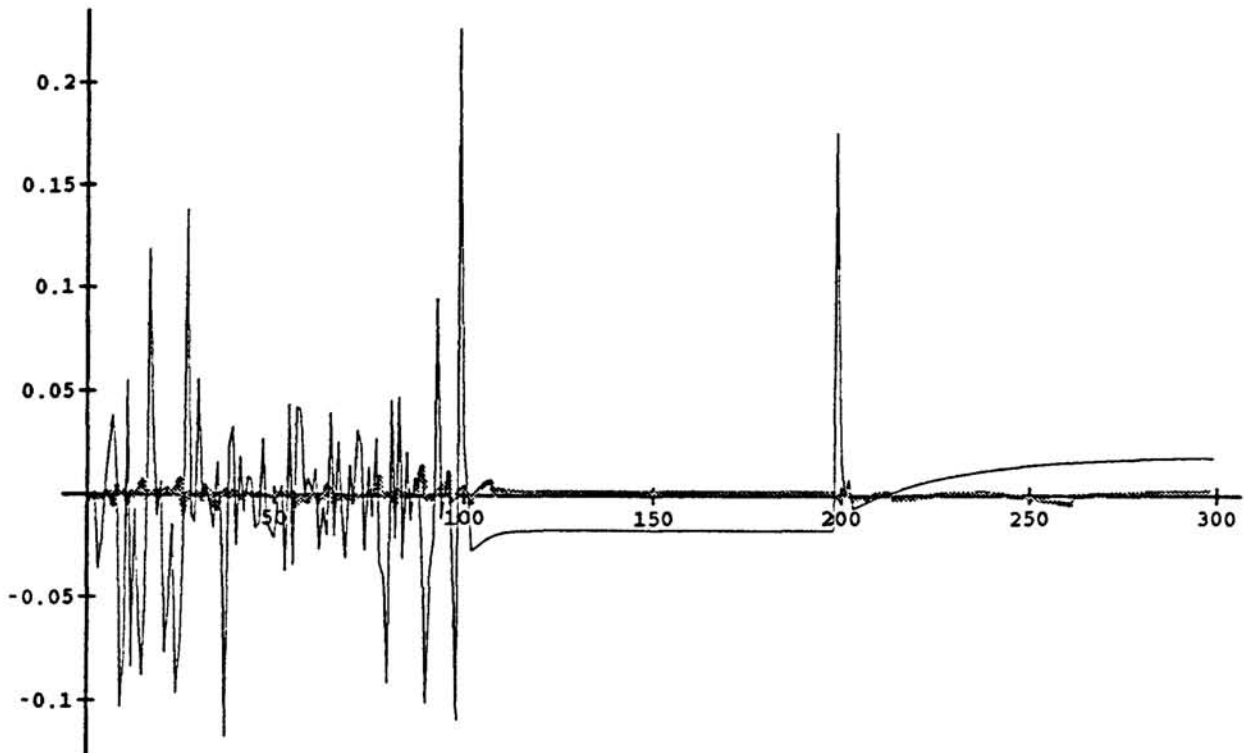

black line : ARMA model estimation error $(y(t + \delta t) - \hat{y}_{ARMA}(t + \delta t))$
grey line : ART2/BP estimation error $(y(t + \delta t) - \hat{y}_{ART2/BP}(t + \delta t))$

Figure 1: Comparison of the estimation error of the ARMA model and the ART2/BP network

To increase the nonlinearities of the transfer function, the area of the tank varies with a step function of the liquid level. The BP subnetworks have 2 input nodes, 1 hidden layer with 2 neurons and a single neuron output layer. In the simulations $\rho = 0.04$ and the last three categorized input/output pairs are stored at every subnetwork. As the input space is 2-dimensional giving 4 neighbouring nodes the maximum size of the training set 7 input/output pairs. After a learning period of 1000 samples with random inlet flow , three test cases are run with the network in estimation mode. The network had then formed about 140 categories. The same set of simulation data is also run through an offline maximum likelihood method to estimate a linear ARMA model of the plant, see (Ljung 1983). /

Figure 1 shows the simulation results of the three test cases where :

samples 1-100 : random input flow.
samples 101-200 : constant input flow at a low level.
samples 201-300 : constant input flow at a high level.

In Figure 1, the estimation errors of the two methods are compared. For the first 100 samples with stochastic input flow, the estimation error variance of the

ART2/BP network is roughly a factor 10 less than that of the ARMA-model. The performance of ART2/BP is also significantly better for the constant input flow cases, here the ARMA model has an error of $\sim 0.02$ while the ART2/BP-error is $\sim 0.002$. The overall improvement in estimation error is a reduction of roughly 0.1 . Also keep in mind that ART2/BP is compared to an offline maximum likelihood method while ART2/BP clearly is an online method. The online version of the maximum likelihood would most probably have given a worse performance than the offline version.

## 9   CONCLUSION/COMMENTS

The proposed ART2/BP neural network architecture offers some unique features compared to backpropagation. It provides incremental learning and can be applied to truly adaptive estimation tasks. In our example it also outperforms a classical maximum likelihood method for the estimation of a discrete dynamic nonlinear transfer function. Future work will be the investigation of ART2/BP's properties for multistep-ahead prediction of dynamic nonlinear transfer functions, and embedding ART2/BP in a neural adaptive controller.

**Acknowledgments**

Special thanks to Steve Lehar at Boston University for providing me with his ART2 simulation program. It proved to be crucial for getting a quick start on ART2 and understanding the concept.

**References**

Carpenter, G.A. & Grossberg, S. (1987). ART2: Self-organization of stable category recognition codes for analog input patterns. *Applied Optics* pp 4919-4930.

Carpenter, G.A. & Grossberg, S. (1988). The ART of adaptive pattern recognition by a self-organizing neural network. *Computer* 21 pp 77-88.

Fahlman, S.E. (1988). Faster-Learning Variations on Back-Propagation: An Empirical Study. *Proceedings of the 1988 Connectionist Models Summer School*. Morgan Kaufmann.

Ljung, L. & Söderstrøm (1983). Theory and practice of recursive identification. *The MIT press*, Cambridge, MA.

Otwell, K. (1990). Incremental backpropagation learning from novelty-based orthogonalization. *Proceedings IJNN90* .

Poggio, T., Girosi, F. (1990). Networks for Approximation and Learning. *Proceedings of the IEEE*,Vol. 78, No. 9.

Rumelhart, D.E., Hinton, G.E., & Williams, R.J. (1986). Parallel Distributed Processing: Explorations in the microstructure of Cognition, Vol. 1. *The MIT Press*,Cambridge, MA.

Ryan, T. W. (1988). The resonance correlation network. *Proceedings IJNN88*.
